# High-Speed Airborne Particle Monitoring Using Artificial Neural Networks

**Alistair Ferguson**
ERDC, Univ. of Hertfordshire
A.Ferguson@herts.ac.uk

**Theo Sabisch**
Dept. Electrical and Electronic Eng.
Univ. of Hertfordshire

**Paul Kaye**
ERDC, Univ. of Hertfordshire

**Laurence C. Dixon**
NOC, Univ. of Hertfordshire

**Hamid Bolouri**
ERDC, Univ. of Hertfordshire, Herts, AL10 9AB, UK

## Abstract

Current environmental monitoring systems assume particles to be spherical, and do not attempt to classify them. A laser-based system developed at the University of Hertfordshire aims at classifying airborne particles through the generation of two-dimensional scattering profiles. The performances of template matching, and two types of neural network (HyperNet and *semi-linear units*) are compared for image classification. The neural network approach is shown to be capable of comparable recognition performance, while offering a number of advantages over template matching.

## 1  Introduction

Reliable identification of low concentrations of airborne particles requires high speed monitoring of large volumes of air, and incurs heavy computational overheads. An instrument to detect particle shape and size from spatial light scattering profiles has

previously been described [6]. The system constrains individual particles to traverse a laser beam. Thus, spatial distributions of the light scattered by individual particles may be recorded as two dimensional grey-scale images.

Due to their highly distributed nature, Artificial Neural Networks (ANNs) offer the possibility of high-speed non-linear pattern classification. Their use in particulate classification has already been investigated. The work by Kohlus [7] used contour data extracted from microscopic images of particles, and so was not real-time. While using laser scattering data to allow real-time analysis, Bevan [2] used only three photomultipliers, from which very little shape information can be collected.

This paper demonstrates the plausibility of particle classification based on shape recognition using an ANN. While capable of similar recognition rates, the neural networks are shown to offer a number of advantages over template matching.

## 2   The HyperNet Architecture

HyperNet is the term used to denote the hardware model of a RAM-based sigma-pi neural architecture developed by Gurney [5]. The architecture is similar in nature to the pRAM of Gorse and Taylor (references in [4]). The amenability of these nodes to hardware realisation has been extensively investigated, leading to custom VLSI implementations of both nodes [3, 4]. Each HyperNet node is termed a multi-cube unit (MCU), and consists of a number of subunits, each with an arbitrary number of inputs. $j$ references the nodes, with $i = 1, \ldots, I^j$ indexing the subunits. $\mu$ denotes the site addresses, and is the set of bit strings $\mu_1, \ldots, \mu_n$ where $n$ denotes the number of inputs to the subunit. $z_c$ refers to the $c^{\text{th}}$ real-valued input, with $z_c \in [0,1]$ and $\hat{z}_c \equiv (1 - z_c)$. For each of the $2^n$ site store locations, two sets are defined: $c \in M_{\mu 0}^{ij}$ if $\mu_c = 0$; $c \in M_{\mu 1}^{ij}$ if $\mu_c = 1$. The access probability $P(\mu^{ij})$ for location $\mu$ in subunit $i$ of hidden layer node $j$ is therefore

$$P(\mu^{ij}) = \prod_{c \in M_{\mu 0}^{ij}} \hat{z}_c \prod_{c \in M_{\mu 1}^{ij}} z_c \tag{1}$$

The activation $(a^j)$ is formed by accumulating the proportional site values $(S_{\mu^{ij}})$ from every subunit. The activation is then passed through a sigmoidal transfer function to yield the node output $(y^j)$.

$$a^j = \frac{1}{I^j} \sum_{i=1}^{I^j} \sum_{\mu^{ij}} S_{\mu^{ij}} P(\mu^{ij}) \tag{2}$$

$$y^j = \sigma(a^j) = \frac{1}{1 + e^{a^j / \rho}} \tag{3}$$

where $\rho$ is a positive parameter determining the steepness of the sigmoidal curve. By combining equations (1) and (2), it becomes apparent that the node is a *higher-order* or *sigma-pi* node [9]. A wide variety of learning algorithms have been tailored for these nodes, notably reward-penalty and back-propagation [5].

## 3    Description of the Particle Monitoring System

The instrument draws air through the laser scattering chamber at approximately $1.5$ min$^{-1}$, and is constrained to a column of approximately 0.8mm diameter at the intersection with the laser beam. Light scattered into angles between 30° and 141° to the beam direction is reflected through the optics and onto the photocathode of an intensified CCD (charge-coupled device), thus giving rise to the scattering profile. The imaging device used has a pixel resolution of $385 \times 288$, which is quantised into $256^2$ 8-bit pixels by the frame grabbing processor card of the host computer.

Data was collected on eight particle types, namely: long and short caffeine fibres; $3\mu$m and $12\mu$m micro-machined silicon dioxide fibres; copper flakes (2–5$\mu$m in length and $0.1\mu$m thick); $3\mu$m and $4.3\mu$m polystyrene spheres; and salt crystals. An exemplar profile for each class is given in figure 1. Almost all the image types are highly variable. In particular, the scattering profile obtained for a fibrous particle is affected by its orientation as it passes through the laser beam. The scattering profiles are intrinsically centred, with the scaling giving important information regarding the size of the particle. The experiments reported here use 100 example scattering profiles for each of the eight particle classes. For each class, 50 randomly selected images were used to construct the templates or train the neural network (training set), and the remainder used to test the performance of the pattern classifiers.

## 4    Experimental Results

The performance of template matching is compared to both HyperNet and networks of semi-linear units. In all experiments, high-speed classification is emphasised by

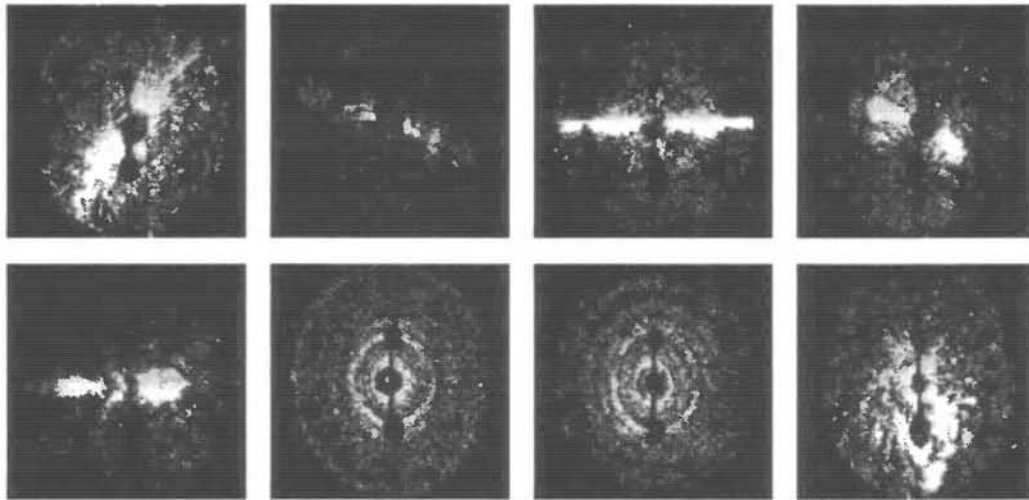

Figure 1: Exemplar Image Profile For Each Of The Eight Benchmark Classes

avoiding image preprocessing operations such as transformation to the frequency domain, histogram equalisation, and other filtering operations. Furthermore, all experiments use the scatter profile image as input, and include no other information.

The current monitoring system produces a $256^2$ 8-bit pixel image. The sensitivity of the camera is such that a single pixel can represent the registration of a single photon of light. Two possible methods of reducing computation, implementable through the use of a cheaper, less sensitive camera were investigated. The first grouped neighbouring pixels to form a single average intensity value. The neighbourhood size was restricted to powers of two, producing images ranging in size from $256^2$ to $4^2$ pixels. The second banded grey levels into groups, again in powers of two. Each pixel could therefore range from eight bits down to one.

## 4.1  Template Matching Results

The construction of reference templates is crucial to successful classification. Two approaches to template construction were investigated

① Single reference image for each class. Various techniques were applied ranging from individual images, to mode, median, and mean averaged templates. Mean averaged templates were found to lead to the highest classification rates. In this approach, each pixel location in the template takes on the averaged value of that location across the 50 training images.

② Multiple templates per class. A K-means clustering algorithm [1] was used to identify clusters of highly correlated images within each class. The initial cluster centres were hand selected. The maximum number of clusters within each class was limited to six. For each cluster, the reference template was constructed using the mean averaging approach above.

Tables 1 and 2 summarise the recognition rates achieved using single, and multiple mean averaged templates for each particle class. In both cases, the best average recognition rate using this approach was gained with $128^2$ 3-bit pixel images. With a single template this lead to a recognition rate of 78.2%, increasing to 85.2% for multiple templates. However, the results for both $16^2$ and $8^2$ pixel images are reasonable approximations of the best performance, and represent an acceptable trade-off between computational cost and performance. With few exceptions, multiple templates per class led to higher recognition rates than for the corresponding single template results. This is attributable to the variability of the particles within a class. As expected, the effect of grey level quantisation is inversely proportional to that of local averaging.

In order to evaluate the efficiency of the template construction methods, every image in the training set was used as a reference template. $256^2$ 8-bit, $128^2$ 3-bit, and $64^2$ 2-bit pixel images were used for these experiments. However, the recognition rate did not exceed 85%, demonstrating the success of the template generation schemes previously employed.

Table 1: Single Template Per Class % Recognition Rates

| grey levels | image size | | | | | | |
|---|---|---|---|---|---|---|---|
| | $256^2$ | $128^2$ | $64^2$ | $32^2$ | $16^2$ | $8^2$ | $4^2$ |
| 256 | 73.5 | 75.0 | 74.7 | 74.7 | 74.7 | 75.0 | 67.2 |
| 128 | 73.5 | 75.0 | 74.7 | 74.7 | 74.5 | 75.0 | **68.5** |
| 64 | 73.0 | 75.0 | 74.5 | 74.5 | 74.2 | 74.7 | 66.2 |
| 32 | 73.0 | 74.7 | 75.2 | 75.5 | 74.7 | 74.2 | 66.5 |
| 16 | 74.0 | 76.0 | 76.7 | 76.0 | 75.0 | **75.5** | 56.0 |
| 8 | **75.5** | **78.2** | **77.5** | **77.5** | **76.0** | 73.7 | 38.7 |
| 4 | 68.4 | 69.7 | 71.0 | 70.7 | 69.7 | 58.5 | 18.7 |
| 2 | 69.7 | 68.7 | 65.5 | 66.2 | 46.2 | 23.0 | 16.6 |

Table 2: Multiple Templates Per Class % Recognition Rates

| grey levels | image size | | | | | | |
|---|---|---|---|---|---|---|---|
| | $256^2$ | $128^2$ | $64^2$ | $32^2$ | $16^2$ | $8^2$ | $4^2$ |
| 256 | 78.0 | 80.0 | 80.2 | 80.5 | 79.0 | 76.7 | **70.2** |
| 128 | 78.5 | 80.2 | 80.5 | 80.5 | 79.0 | 77.0 | 69.7 |
| 64 | 78.7 | 80.2 | 80.2 | 80.5 | 79.2 | 76.0 | 69.2 |
| 32 | 78.2 | 81.2 | 81.7 | 80.0 | 78.7 | 76.7 | 67.7 |
| 16 | 80.2 | 83.5 | 83.0 | 81.2 | 79.5 | 78.5 | 56.0 |
| 8 | **82.2** | **85.2** | **84.5** | **84.7** | **81.0** | **80.0** | 43.5 |
| 4 | 72.7 | 74.5 | 72.2 | 72.2 | 69.5 | 61.2 | 39.2 |
| 2 | 69.7 | 70.2 | 70.7 | 62.7 | 51.7 | 51.7 | 0.03 |

## 4.2   Neural Network Results

A fully connected three layer feed-forward network was used in all experiments. The number of hidden layer neurons was equal to the square root of the number of pixels. The target patterns were chosen to minimise the number of output layer nodes, while ensuring an equitable distribution of zeros and ones. Six output layer neurons were used to give a minimum Hamming distance of two between target patterns. The classification of a pattern was judged to be the particle class whose target pattern was closest (lowest difference error). The HyperNet architecture was trained using steepest descent, though the line search was hardware based and inexact. The semi-linear network was trained using a variety of back-propagation type algorithms, with the best results obtained reported. Both networks were randomly initialised. Due to the enormous training overhead, only $16^2$ and $8^2$ pixel images were tried. The recognition rates achieved are given in table 3.

Both neural networks are significantly better than the single, and some of the multiple template matching results. With optimisation of the network structures, it is likely that the ANNs could exceed the performance of multiple templates.

Table 3: Neural Network % Recognition Rates

| Classifier | Quantisation Levels | | | |
|---|---|---|---|---|
| | $16^2$ 4 bit | $16^2$ 3-bit | $8^2$ 4-bit | $8^2$ 3-bit |
| HyperNet | 83.8 | 82.3 | 83.0 | 76.8 |
| Semi-linear | 84.5 | 86.3 | 77.8 | 76.0 |

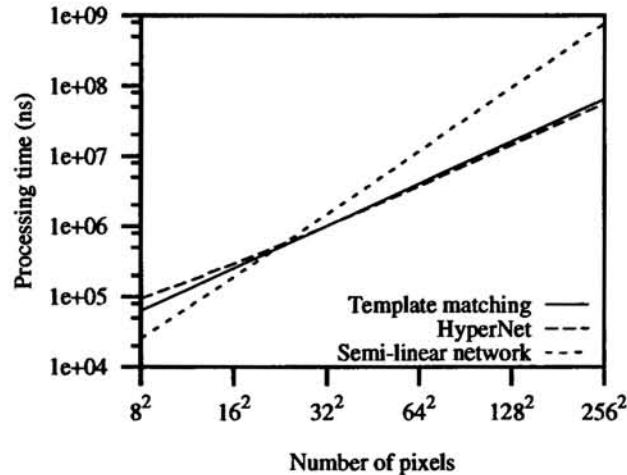

Figure 2: Hardware classification speeds for a single pattern against image size

## 5  Speed Considerations

Single processor, pipelined hardware implementations of the three classification techniques have been considered. A fast (45ns) multiply-accumulate chip (Logic Devices Ltd, LMA2010) was utilised for semi-linear units. Both template matching and HyperNet were implemented using the Logic Devices LGC381 ALU (26ns per accumulate). The cost of these devices is approximately the same (£10–20). The HyperNet implementation uses a bit-stream approach to eliminate the probability multiplications [8], with a stream length of 256 bits. Figure 2 plots single pattern processing time for each classifier against image size.

For small image resolutions, the semi-linear network offers the best performance, being almost three times faster than template matching. However, template matching and HyperNet yield faster performance at higher image resolutions. At the optimum (indicated by template matching results (§4.1); $128^2$ pixels), HyperNet is almost seven times faster than the comparable implementation of semi-linear units. While the hardware performance of template matching is similar to HyperNet, it suffers from a number of disadvantages to which the neural approaches are immune

① Recognition rate is dependent on the choice of reference images.

② Multiple reference images must be used to achieve good recognition rates

which drastically increases the amount of computation required.

③ New reference images must be found whenever a new class is introduced.

④ Difficult to make behaviour adaptive, ie. respond to changing conditions.

## 6  Conclusions

The feasibility of constructing an airborne particle monitoring system capable of reliable particle identification at high speeds has been demonstrated. Template matching requires multiple reference images and is cumbersome to develop. The neural networks offer easier training procedures and equivalent recognition rates. In addition, HyperNet has the advantage of high speed operation at large image sizes.

### Acknowledgements

The authors would like to thank Dr. Eric Dykes and Dr. Edwin Hirst at the University of Hertfordshire, Dr. Kevin Gurney at Brunel University, and the EPSRC and the Royal Society for financial support.

## References

[1] Stephen Banks. *Signal Processing, Image Processing, and Pattern Recognition.* Prentice Hall, 1990.

[2] A V Bevan et al. The application of neural networks to particle shape classification. *Journal of Aerosol Science*, 23(Suppl. 1):329–332, 1992.

[3] Hamid Bolouri et al. Design, manufacture, and evaluation of a scalable high-performance neural system. *Electronics Letters*, 30(5):426–427, 3 March 1994.

[4] T G Clarkson et al. The pRAM: An adaptive VLSI chip. *IEEE Transactions on Neural Networks*, 4(3):408–412, May 1993.

[5] Kevin N Gurney. *Learning in networks of structured hypercubes.* PhD thesis, Department of Electrical Engineering, UK, 1995.

[6] Paul H Kaye et al. Airborne particle shape and size classification from spatial light scattering profiles. *Journal of Aerosol Science*, 23(6):597–611, 1992.

[7] R Kohlus et al. Particle shape analysis as an example of knowledge extraction by neural nets. *Part. Part. Syst. Charact.*, 10:275–278, 1993.

[8] Paul Morgan et al. Hardware implementation of a real-valued sigma-pi network. In *Artificial Neural Networks 5*, volume 2, pages 351–356, North-Holland, 1995.

[9] David E Rumelhart et al. *Parallel Distributed Processing: Explorations in the Macrostructure of Cognition*, volume 1. MIT Press, 1986.
